# Information Geometrical Framework for Analyzing Belief Propagation Decoder

**Shiro Ikeda**
Kyushu Inst. of Tech., & PRESTO, JST
Wakamatsu, Kitakyushu, Fukuoka, 808-0196 Japan
*shiro@brain.kyutech.ac.jp*

**Toshiyuki Tanaka**
Tokyo Metropolitan Univ.
Hachioji, Tokyo, 192-0397 Japan
*tanaka@eei.metro-u.ac.jp*

**Shun-ichi Amari**
RIKEN BSI
Wako, Saitama, 351-0198 Japan
*amari@brain.riken.go.jp*

## Abstract

The mystery of belief propagation (BP) decoder, especially of the turbo decoding, is studied from information geometrical viewpoint. The loopy belief network (BN) of turbo codes makes it difficult to obtain the true "belief" by BP, and the characteristics of the algorithm and its equilibrium are not clearly understood. Our study gives an intuitive understanding of the mechanism, and a new framework for the analysis. Based on the framework, we reveal basic properties of the turbo decoding.

## 1 Introduction

Since the proposal of turbo codes[2], they have been attracting a lot of interests because of their high performance of error correction. Although the thorough experimental results strongly support the potential of this iterative decoding method, the mathematical background is not sufficiently understood. McEliece et al.[5] have shown its relation to the Pearl's BP, but the BN for the turbo decoding is loopy, and the BP solution gives only an approximation.

The problem of the turbo decoding is a specific example of a general problem of marginalizing an exponential family distribution. The distribution includes higher order correlations, and its direct marginalization is intractable. But the partial model with a part of the correlations, can be marginalized with BP algorithm exactly, since it does not have any loop. By collecting and exchanging the BP results of the partial models, the true "belief" is approximated. This structure is common among various iterative methods, such as Gallager codes, Bethé approximation in statistical physics[4], and BP for loopy BN.

We investigate the problem from information geometrical viewpoint[1]. It gives a new framework for analyzing these iterative methods, and shows an intuitive understanding of them. Also it reveals a lot of basic properties, such as characteristics of the equilibrium, the condition of stability, the cost function related to the decoder, and the decoding error. In this

paper, we focus on the turbo decoding, because its structure is simple, but the framework is general, and the main results can be generalized.

## 2   Information Geometrical Framework

### 2.1   Marginalization, MPM Decoding, and Belief

Let us consider a distribution of $\boldsymbol{x} = (x_1, \cdots, x_N)^T$ which is defined as follows

$$p(\boldsymbol{x}) = C \exp(c_0(\boldsymbol{x}) + c_1(\boldsymbol{x}) + \cdots + c_K(\boldsymbol{x})), \qquad (1)$$

where, $c_0(\boldsymbol{x})$ is the linear function of $\{x_i\}$, and each $c_k(\boldsymbol{x})$ is the higher order correlations of $\{x_i\}$. The problem of turbo codes and similar iterative methods are to marginalize this distribution. Let $\Pi$ denote the operator of marginalization as, $\Pi \circ p(\boldsymbol{x}) \overset{\mathrm{def}}{=} \prod_{i=1}^N p(x_i)$. The marginalization is equivalent to take the expectation of $\boldsymbol{x}$ as

$$\boldsymbol{\eta} \overset{\mathrm{def}}{=} \sum_{\boldsymbol{x}} \boldsymbol{x} p(\boldsymbol{x}), \quad \boldsymbol{\eta} = (\eta_1, \cdots, \eta_N)^T.$$

In the case of MPM (maximization of the posterior marginals) decoding, $x_i \in \{-1, +1\}$ and the sign of each $\eta_i$ is the decoding result. In the belief network, $x_i \in \{0, 1\}$ and $\eta_i$ is the belief. In these iterative methods, the marginalization of eq.(1) is not tractable, but the marginalization of the following distribution is tractable.

$$p_r(\boldsymbol{x}; \boldsymbol{\xi}) = \exp\left(c_0(\boldsymbol{x}) + c_r(\boldsymbol{x}) + \boldsymbol{\xi} \cdot \boldsymbol{x} - \varphi_r(\boldsymbol{\xi})\right), \quad r = 1, \cdots, K, \quad \boldsymbol{\xi} \in \mathcal{R}^N. \qquad (2)$$

Each $p_r(\boldsymbol{x}; \boldsymbol{\xi})$ includes only one of the $\{c_k(\boldsymbol{x})\}$ in eq.(1), and additional parameter $\boldsymbol{\xi}$ is used to adjust linear part of $\boldsymbol{x}$. The iterative methods are exchanging information through $\boldsymbol{\xi}$ for each $p_r$, and finally approximate $\Pi \circ p(\boldsymbol{x})$.

### 2.2   The Case of Turbo Decoding

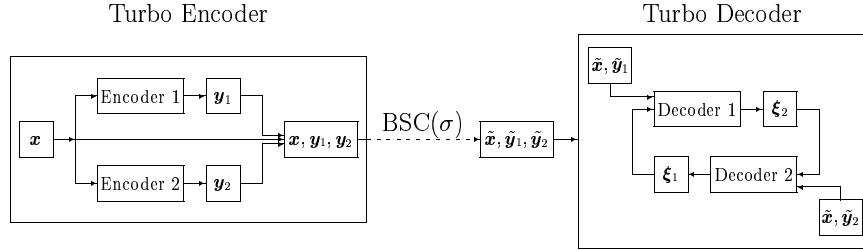

Figure 1: Turbo codes

In the case of turbo codes, $\boldsymbol{x}$ is the information bits, from which the turbo encoder generates two sets of parity bits, $\boldsymbol{y}_1 = (y_{11}, \cdots, y_{1L})^T$, and $\boldsymbol{y}_2 = (y_{21}, \cdots, y_{2L})^T$, $y_{1j}, y_{2j} \in \{-1, +1\}$ (Fig.1). Each parity bit is expressed as the form $\prod_i x_i$, where the product is taken over a subset of $\{1, \cdots, N\}$. The codeword $(\boldsymbol{x}, \boldsymbol{y}_1, \boldsymbol{y}_2)$ is then transmitted over a noisy channel, which we assume BSC (binary symmetric channel) with flipping probability $\sigma < 1/2$. The receiver observes $(\tilde{\boldsymbol{x}}, \tilde{\boldsymbol{y}}_1, \tilde{\boldsymbol{y}}_2)$, $\tilde{x}_i, \tilde{y}_{1j}, \tilde{y}_{2j} \in \{-1, +1\}$.

The ultimate goal of the turbo decoding is the MPM decoding of $\boldsymbol{x}$ based on $p(\boldsymbol{x} | \tilde{\boldsymbol{x}}, \tilde{\boldsymbol{y}}_1, \tilde{\boldsymbol{y}}_2)$. Since the channel is memoryless, the following relation holds

$$p(\tilde{\boldsymbol{x}}, \tilde{\boldsymbol{y}}_1, \tilde{\boldsymbol{y}}_2 | \boldsymbol{x}) = \exp\left(\beta \tilde{\boldsymbol{x}} \cdot \boldsymbol{x} + \beta \tilde{\boldsymbol{y}}_1 \cdot \boldsymbol{y}_1 + \beta \tilde{\boldsymbol{y}}_2 \cdot \boldsymbol{y}_2 - (N + 2L)\psi(\beta)\right)$$

$$\beta > 0, \quad \sigma = \frac{1}{2}(1 - \tanh \beta), \quad \psi(\beta) \overset{\mathrm{def}}{=} \ln(e^\beta + e^{-\beta}).$$

By assuming the uniform prior on $\boldsymbol{x}$, the posterior distribution is given as follows

$$p(\boldsymbol{x}|\tilde{\boldsymbol{x}},\tilde{\boldsymbol{y}}_1,\tilde{\boldsymbol{y}}_2) = \frac{p(\tilde{\boldsymbol{x}},\tilde{\boldsymbol{y}}_1,\tilde{\boldsymbol{y}}_2|\boldsymbol{x})}{\sum_{\boldsymbol{x}} p(\tilde{\boldsymbol{x}},\tilde{\boldsymbol{y}}_1,\tilde{\boldsymbol{y}}_2|\boldsymbol{x})} = C \exp\left(\beta\tilde{\boldsymbol{x}}\cdot\boldsymbol{x} + \beta\tilde{\boldsymbol{y}}_1\cdot\boldsymbol{y}_1 + \beta\tilde{\boldsymbol{y}}_2\cdot\boldsymbol{y}_2\right)$$
$$= C \exp\left(c_0(\boldsymbol{x}) + c_1(\boldsymbol{x}) + c_2(\boldsymbol{x})\right). \tag{3}$$

Here $C$ is the normalizing factor, and $c_0(\boldsymbol{x}) = \beta\tilde{\boldsymbol{x}}\cdot\boldsymbol{x}$, $c_r(\boldsymbol{x}) = \beta\tilde{\boldsymbol{y}}_r\cdot\boldsymbol{y}_r$ $(r = 1,2)$. Equation(3) is equivalent to eq.(1), where $K = 2$. When $N$ is large, marginalization of $p(\boldsymbol{x}|\tilde{\boldsymbol{x}},\tilde{\boldsymbol{y}}_1,\tilde{\boldsymbol{y}}_2)$ is intractable since it needs summation over $2^N$ terms. Turbo codes utilize two decoders which solve the MPM decoding of $p_r(\boldsymbol{x};\boldsymbol{\xi})$ $(r = 1,2)$ in eq.(2). The distribution is derived from $p(\tilde{\boldsymbol{x}},\tilde{\boldsymbol{y}}_r|\boldsymbol{x})$ and the prior of $\boldsymbol{x}$ which has the form of

$$\omega(\boldsymbol{x};\boldsymbol{\xi}) = \exp(\boldsymbol{\xi}\cdot\boldsymbol{x} - \psi(\boldsymbol{\xi})).$$

$\omega(\boldsymbol{x};\boldsymbol{\xi})$ is a factorizable distribution. The marginalization of $p(\tilde{\boldsymbol{x}},\tilde{\boldsymbol{y}}_r|\boldsymbol{x})$ is feasible since its BN is loop free. The parameter $\boldsymbol{\xi}$ serves as the window of exchanging the information between the two decoders. The MPM decoding is approximated by updating $\boldsymbol{\xi}$ iteratively in "turbo" like way.

### 2.3 Information Geometrical View of MPM Decoding

Let us consider the family of all the probability distributions over $\boldsymbol{x}$. We denote it by $S$, which is defined as

$$S = \left\{ p(\boldsymbol{x}) \middle| p(\boldsymbol{x}) > 0, \boldsymbol{x} \in \{-1,+1\}^N, \sum_{\boldsymbol{x}} p(\boldsymbol{x}) = 1 \right\}.$$

We consider an $e$–flat submanifold $M_0$ in $S$. This is the submanifold of $p_0(\boldsymbol{x};\boldsymbol{\theta})$ defined as

$$M_0 = \left\{ p_0(\boldsymbol{x};\boldsymbol{\theta}) = \exp\left(c_0(\boldsymbol{x}) + \boldsymbol{\theta}\cdot\boldsymbol{x} - \varphi_0(\boldsymbol{\theta})\right) \middle| \boldsymbol{\theta} = (\theta^1,\cdots,\theta^N)^T \in \mathcal{R}^N \right\}. \tag{4}$$

Since $c_0(\boldsymbol{x}) = \beta\tilde{\boldsymbol{x}}\cdot\boldsymbol{x}$, every distribution of $M_0$ can be rewritten as follows

$$p_0(\boldsymbol{x};\boldsymbol{\theta}) = \exp\left(c_0(\boldsymbol{x}) + \boldsymbol{\theta}\cdot\boldsymbol{x} - \varphi_0(\boldsymbol{\theta})\right) = \exp\left((\beta\tilde{\boldsymbol{x}} + \boldsymbol{\theta})\cdot\boldsymbol{x} - \varphi_0(\boldsymbol{\theta})\right).$$

It shows that every distribution of $M_0$ is decomposable, or factorizable. From the information geometry[1], we have the following theorem of $m$–projection.

**Theorem 1.** *Let $M$ be an $e$–flat submanifold in $S$, and let $q(\boldsymbol{x}) \in S$. The point in $M$ that minimizes the KL-divergence from $q(\boldsymbol{x})$ to $M$, is denoted by,*

$$\Pi_M \circ q(\boldsymbol{x}) = \underset{p(\boldsymbol{x}) \in M}{\operatorname{argmin}} D[q(\boldsymbol{x});p(\boldsymbol{x})],$$

*and is called the $m$–projection of $q(\boldsymbol{x})$ to $M$. The $m$–projection is unique.* $\square$

It is easy to show that the marginalization corresponds to the $m$–projection to $M_0$[7]. Since MPM decoding and marginalization is equivalent, MPM decoding is also equivalent to the $m$–projection to $M_0$.

### 2.4 Information Geometry of Turbo Decoding

Let $\pi_M \circ q(\boldsymbol{x})$ denote the parameters in $M$ of the $m$–projected distribution,

$$\pi_M \circ q(\boldsymbol{x}) = \underset{\boldsymbol{\theta} \in \mathcal{R}^N}{\operatorname{argmin}} D[q(\boldsymbol{x});p(\boldsymbol{x};\boldsymbol{\theta})].$$

The turbo decoding process is written as follows,

1. Let $\boldsymbol{\xi}_1^t = 0$ for $t = 0$, and $t = 1$.

2. Project $p_2(\boldsymbol{x}; \boldsymbol{\xi}_1^t)$ onto $M_0$ as $\boldsymbol{\theta} = \pi_{M_0} \circ p_2(\boldsymbol{x}; \boldsymbol{\xi}_1^t)$, and calculate $\boldsymbol{\xi}_2^{t+1}$ by
$$\boldsymbol{\xi}_2^{t+1} = \pi_{M_0} \circ p_2(\boldsymbol{x}; \boldsymbol{\xi}_1^t) - \boldsymbol{\xi}_1^t.$$

3. Project $p_1(\boldsymbol{x}; \boldsymbol{\xi}_2^{t+1})$ onto $M_0$ as $\boldsymbol{\theta} = \pi_{M_0} \circ p_1(\boldsymbol{x}; \boldsymbol{\xi}_2^{t+1})$, and calculate $\boldsymbol{\xi}_1^{t+1}$ by
$$\boldsymbol{\xi}_1^{t+1} = \pi_{M_0} \circ p_1(\boldsymbol{x}; \boldsymbol{\xi}_2^{t+1}) - \boldsymbol{\xi}_2^{t+1}.$$

4. If $\pi_{M_0} \circ p_1(\boldsymbol{x}; \boldsymbol{\xi}_2^{t+1}) \neq \pi_{M_0} \circ p_2(\boldsymbol{x}; \boldsymbol{\xi}_1^{t+1})$, go to step 2.

The turbo decoding approximates the estimated parameter $\boldsymbol{\theta}^*$, the projection of $p(\boldsymbol{x}|\tilde{\boldsymbol{x}}, \tilde{\boldsymbol{y}}_1, \tilde{\boldsymbol{y}}_2)$ onto $M_0$, as $\boldsymbol{\theta}^* = \boldsymbol{\xi}_1^* + \boldsymbol{\xi}_2^*$, where the estimated distribution is

$$p_0(\boldsymbol{x}; \boldsymbol{\theta}^*) = \exp\left(c_0(\boldsymbol{x}) + \boldsymbol{\xi}_1^* \cdot \boldsymbol{x} + \boldsymbol{\xi}_2^* \cdot \boldsymbol{x} - \varphi_0(\boldsymbol{\xi}_1^* + \boldsymbol{\xi}_2^*)\right). \tag{5}$$

An intuitive understanding of the turbo decoding is as follows. In step 2, $(\boldsymbol{\xi}_2^* \cdot \boldsymbol{x})$ in eq.(5) is replaced with $c_2(\boldsymbol{x})$. The distribution becomes $p_2(\boldsymbol{x}; \boldsymbol{\xi}_1^*)$, and $\boldsymbol{\xi}_2^*$ is estimated by projecting it onto $M_0$. In step 3, $(\boldsymbol{\xi}_1^* \cdot \boldsymbol{x})$ in eq.(5) is replaced with $c_1(\boldsymbol{x})$, and $\boldsymbol{\xi}_1^*$ is estimated by $m-$projection of $p_1(\boldsymbol{x}; \boldsymbol{\xi}_2^*)$.

We now define the submanifold corresponding to each decoder,

$$M_r = \left\{ p_r(\boldsymbol{x}; \boldsymbol{\xi}) = \exp\left(c_0(\boldsymbol{x}) + c_r(\boldsymbol{x}) + \boldsymbol{\xi} \cdot \boldsymbol{x} - \varphi_r(\boldsymbol{\xi})\right) | \boldsymbol{\xi} = (\xi^1, \cdots, \xi^N)^T \in \mathcal{R}^N \right\}$$
$$r = 1, 2.$$

$\boldsymbol{\xi}$ is the coordinate system of $M_r$. $M_r$ is also an $e$–flat submanifold. $M_1 \neq M_2$ and $M_r \neq M_0$ hold because $c_r(\boldsymbol{x})$ includes cross terms of $\boldsymbol{x}$ and $c_1(\boldsymbol{x}) \neq c_2(\boldsymbol{x})$ in general. The information geometrical view of the turbo decoding is schematically shown in Fig.2.

## 3 The Properties of Belief Propagation Decoder

### 3.1 Equilibrium

When the the turbo decoding converges, equilibrium solution defines three important distributions, $p_1(\boldsymbol{x}; \boldsymbol{\xi}_1^*)$, $p_2(\boldsymbol{x}; \boldsymbol{\xi}_2^*)$, and $p_0(\boldsymbol{x}; \boldsymbol{\theta}^*)$. They satisfy the following two conditions:

1. $\Pi \circ p_1(\boldsymbol{x}; \boldsymbol{\xi}_2^*) = \Pi \circ p_2(\boldsymbol{x}; \boldsymbol{\xi}_1^*) = p_0(\boldsymbol{x}; \boldsymbol{\theta}^*)$. \quad (6)

2. $\boldsymbol{\theta}^* = \boldsymbol{\xi}_1^* + \boldsymbol{\xi}_2^*$. \quad (7)

Let us define a manifold $M(\boldsymbol{\theta})$ as

$$M(\boldsymbol{\theta}) = \left\{ p(\boldsymbol{x}) \Big| p(\boldsymbol{x}) \in S, \ \sum_{\boldsymbol{x}} p(\boldsymbol{x})\boldsymbol{x} = \sum_{\boldsymbol{x}} p_0(\boldsymbol{x}; \boldsymbol{\theta})\boldsymbol{x} \right\}.$$

From its definition, for any $p(\boldsymbol{x}) \in M(\boldsymbol{\theta})$, the expectation of $\boldsymbol{x}$ is the same, and its $m-$projection to $M_0$ coincides with $p_0(\boldsymbol{x}; \boldsymbol{\theta})$. This is an $m$–flat submanifold[1], and we call $M(\boldsymbol{\theta})$ an equimarginal submanifold. Since eq.(6) holds, $p_0(\boldsymbol{x}; \boldsymbol{\theta}^*), p_1(\boldsymbol{x}; \boldsymbol{\xi}_2^*), p_2(\boldsymbol{x}; \boldsymbol{\xi}_1^*) \in M(\boldsymbol{\theta}^*)$ is satisfied.

Let us define an $e$–flat version of the submanifold as $E(\boldsymbol{\theta}^*)$, which connects $p_0(\boldsymbol{x}; \boldsymbol{\theta}^*)$, $p_1(\boldsymbol{x}; \boldsymbol{\xi}_2^*)$, and $p_2(\boldsymbol{x}; \boldsymbol{\xi}_1^*)$ in log-linear manner

$$E(\boldsymbol{\theta}^*) = \left\{ p(\boldsymbol{x}) = C p_0(\boldsymbol{x}; \boldsymbol{\theta}^*)^{t_0} p_1(\boldsymbol{x}; \boldsymbol{\xi}_2^*)^{t_1} p_2(\boldsymbol{x}; \boldsymbol{\xi}_1^*)^{t_2} \Big| \sum_{r=0}^{2} t_r = 1 \right\}.$$

Since eq.(7) holds, $p(\boldsymbol{x}|\tilde{\boldsymbol{x}}, \tilde{\boldsymbol{y}}_1, \tilde{\boldsymbol{y}}_2)$ is included in the $E(\boldsymbol{\theta})$. It can be proved by taking $t_0 = -1, t_1 = t_2 = 1$.

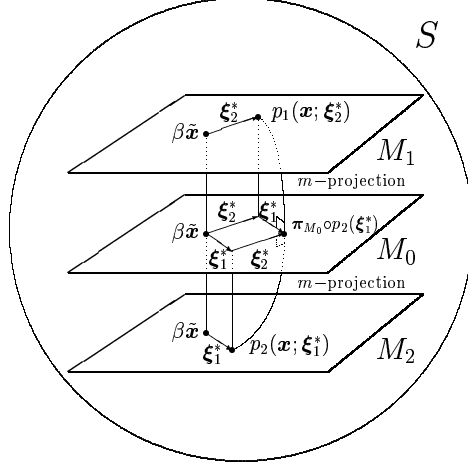

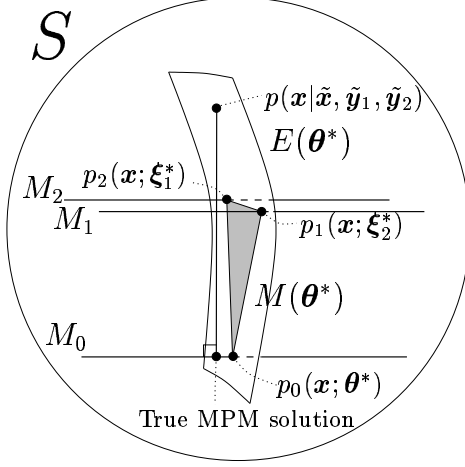

Figure 2: Turbo decoding

Figure 3: $M(\boldsymbol{\theta}^*)$ and $E(\boldsymbol{\theta}^*)$

**Theorem 2.** *When the turbo decoding procedure converges, the convergent probability distributions $p_0(\boldsymbol{x}; \boldsymbol{\theta}^*)$, $p_1(\boldsymbol{x}; \boldsymbol{\xi}_2^*)$, and $p_2(\boldsymbol{x}; \boldsymbol{\xi}_1^*)$ belong to equimarginal submanifold $M(\boldsymbol{\theta}^*)$, while its e–flat version $E(\boldsymbol{\theta}^*)$ includes these three distributions and also the posterior distribution $p(\boldsymbol{x}|\tilde{\boldsymbol{x}}, \tilde{\boldsymbol{y}}_1, \tilde{\boldsymbol{y}}_2)$ (Fig.3).* $\square$

If $M(\boldsymbol{\theta}^*)$ includes $p(\boldsymbol{x}|\tilde{\boldsymbol{x}}, \tilde{\boldsymbol{y}}_1, \tilde{\boldsymbol{y}}_2)$, $p(\boldsymbol{x}; \boldsymbol{\theta}^*)$ is the true marginalization of $p(\boldsymbol{x}|\tilde{\boldsymbol{x}}, \tilde{\boldsymbol{y}}_1, \tilde{\boldsymbol{y}}_2)$. However, $M(\boldsymbol{\theta}^*)$ does not necessarily include $p(\boldsymbol{x}|\tilde{\boldsymbol{x}}, \tilde{\boldsymbol{y}}_1, \tilde{\boldsymbol{y}}_2)$. This fact means that $p(\boldsymbol{x}|\tilde{\boldsymbol{x}}, \tilde{\boldsymbol{y}}_1, \tilde{\boldsymbol{y}}_2)$ and $p_0(\boldsymbol{x}; \boldsymbol{\theta}^*)$ are not necessarily equimarginal, which is the origin of the decoding error.

### 3.2 Condition of Stability

The expectation parameters are defined as follows with $\varphi_0$ in eq.(4) and $\varphi_r$ in eq.(2)

$$\boldsymbol{\eta}_0(\boldsymbol{\theta}) \overset{\text{def}}{=} \sum_{\boldsymbol{x}} \boldsymbol{x} p_0(\boldsymbol{x}; \boldsymbol{\theta}) = \partial_{\boldsymbol{\theta}} \varphi_0(\boldsymbol{\theta}), \quad \boldsymbol{\eta}_r(\boldsymbol{\xi}) \overset{\text{def}}{=} \sum_{\boldsymbol{x}} \boldsymbol{x} p_r(\boldsymbol{x}; \boldsymbol{\xi}) = \partial_{\boldsymbol{\xi}} \varphi_r(\boldsymbol{\xi}) \quad r = 1, 2.$$

Equation (6) is rewritten as follows with these parameters,

$$\boldsymbol{\eta}_0(\boldsymbol{\theta}^*) = \boldsymbol{\eta}_1(\boldsymbol{\xi}_2^*) = \boldsymbol{\eta}_2(\boldsymbol{\xi}_1^*).$$

We give a sufficiently small perturbation $\boldsymbol{\delta}$ to $\boldsymbol{\xi}_1^*$ and apply one turbo decoding step. The $m$–projection from $p_2(\boldsymbol{x}; \boldsymbol{\xi}^* + \boldsymbol{\delta})$ to $M_0$ gives,

$$\boldsymbol{\eta}_0(\boldsymbol{\theta}^* + \Delta\boldsymbol{\theta}) = \boldsymbol{\eta}_2(\boldsymbol{\xi}_1^* + \boldsymbol{\delta})$$
$$\Delta\boldsymbol{\theta} = G_0(\boldsymbol{\theta}^*)^{-1} G_2(\boldsymbol{\xi}_1^*) \boldsymbol{\delta}.$$

Here, $G_0(\boldsymbol{\theta})$ is the Fisher information matrix of $p_0(\boldsymbol{x}; \boldsymbol{\theta})$, and $G_r(\boldsymbol{\xi})$ is that of $p_r(\boldsymbol{x}; \boldsymbol{\xi})$, $(r = 1, 2)$. Note that $G_0(\boldsymbol{\theta})$ is a diagonal matrix. The Fisher information matrix is defined as follows

$$G_0(\boldsymbol{\theta}) = \partial_{\boldsymbol{\theta}\boldsymbol{\theta}'} \varphi_0(\boldsymbol{\theta}) = \partial_{\boldsymbol{\theta}} \boldsymbol{\eta}_0(\boldsymbol{\theta}), \quad G_r(\boldsymbol{\xi}) = \partial_{\boldsymbol{\xi}\boldsymbol{\xi}'} \varphi_r(\boldsymbol{\xi}) = \partial_{\boldsymbol{\xi}} \boldsymbol{\eta}_r(\boldsymbol{\xi}), \quad r = 1, 2.$$

$\boldsymbol{\xi}_2$ in step 2 will be,

$$\boldsymbol{\xi}_2 = \boldsymbol{\xi}_2^* + \left( G_0(\boldsymbol{\theta}^*)^{-1} G_2(\boldsymbol{\xi}_1^*) - I_N \right) \boldsymbol{\delta}.$$

Here, $I_N$ is an identity matrix of size $N$. Following the same line for step 3, we derive the theorem which coincides with the result of Richardson[6].

**Theorem 3.** *Let $\lambda_i$ be the eigenvalues of the matrix $\mathcal{T}$ defined as*

$$\mathcal{T} = \left(G_0(\boldsymbol{\theta}^*)^{-1}G_1(\boldsymbol{\xi}_2^*) - I_N\right)\left(G_0(\boldsymbol{\theta}^*)^{-1}G_2(\boldsymbol{\xi}_1^*) - I_N\right).$$

*When $|\lambda_i| < 1$ holds for all $i$, the equilibrium point is stable.* $\qquad\square$

### 3.3 Cost Function and Characteristics of Equilibrium

We give the cost function which plays an important role in turbo decoding.

$$\mathcal{F}(\boldsymbol{\xi}_1, \boldsymbol{\xi}_2) = \varphi_0(\boldsymbol{\theta}) - (\varphi_1(\boldsymbol{\xi}_2) + \varphi_2(\boldsymbol{\xi}_1)).$$

Here, $\boldsymbol{\theta} = \boldsymbol{\xi}_1 + \boldsymbol{\xi}_2$. This function is identical to the "free energy" defined in [4].

**Theorem 4.** *The equilibrium state $\boldsymbol{\xi}_1^*, \ldots, \boldsymbol{\xi}_K^*$ is the critical point of $\mathcal{F}$.*

*Proof.* Direct calculation gives $\partial_{\boldsymbol{\xi}_1}\mathcal{F} = \boldsymbol{\eta}_0(\boldsymbol{\theta}) - \boldsymbol{\eta}_2(\boldsymbol{\xi}_1)$, $\partial_{\boldsymbol{\xi}_2}\mathcal{F} = \boldsymbol{\eta}_0(\boldsymbol{\theta}) - \boldsymbol{\eta}_1(\boldsymbol{\xi}_2)$. For the equilibrium, $\boldsymbol{\eta}_0(\boldsymbol{\theta}^*) = \boldsymbol{\eta}_1(\boldsymbol{\xi}_2^*) = \boldsymbol{\eta}_2(\boldsymbol{\xi}_1^*)$ holds, and the proof is completed. $\qquad\square$

When $(\boldsymbol{\xi}_r^{t+1} - \boldsymbol{\xi}_r^t)$ is small,

$$\begin{pmatrix}\boldsymbol{\xi}_1^{t+1}\\ \boldsymbol{\xi}_2^{t+1}\end{pmatrix} - \begin{pmatrix}\boldsymbol{\xi}_1^{t}\\ \boldsymbol{\xi}_2^{t}\end{pmatrix} \simeq -\begin{pmatrix}O & G_0(\boldsymbol{\theta})^{-1}\\ G_0(\boldsymbol{\theta})^{-1} & O\end{pmatrix}\begin{pmatrix}\partial_{\boldsymbol{\xi}_1}\mathcal{F}\\ \partial_{\boldsymbol{\xi}_2}\mathcal{F}\end{pmatrix}.$$

This shows how the algorithm works, but it does not give the characteristics of the equilibrium point. The Hessian of $\mathcal{F}$ is

$$\mathcal{H} = \begin{pmatrix}\partial_{\boldsymbol{\xi}_1\boldsymbol{\xi}_1}\mathcal{F} & \partial_{\boldsymbol{\xi}_1\boldsymbol{\xi}_2}\mathcal{F}\\ \partial_{\boldsymbol{\xi}_2\boldsymbol{\xi}_1}\mathcal{F} & \partial_{\boldsymbol{\xi}_2\boldsymbol{\xi}_2}\mathcal{F}\end{pmatrix} = \begin{pmatrix}G_0 - G_1 & G_0\\ G_0 & G_0 - G_2\end{pmatrix}.$$

And by transforming the variables as, $\boldsymbol{\theta} = \boldsymbol{\xi}_1 + \boldsymbol{\xi}_2$ and $\boldsymbol{\nu} = \boldsymbol{\xi}_1 - \boldsymbol{\xi}_2$, we have

$$\begin{pmatrix}\partial_{\boldsymbol{\theta}\boldsymbol{\theta}}\mathcal{F} & \partial_{\boldsymbol{\theta}\boldsymbol{\nu}}\mathcal{F}\\ \partial_{\boldsymbol{\nu}\boldsymbol{\theta}}\mathcal{F} & \partial_{\boldsymbol{\nu}\boldsymbol{\nu}}\mathcal{F}\end{pmatrix} = \frac{1}{4}\begin{pmatrix}4G_0(\boldsymbol{\theta}) - (G_1 + G_2) & (G_1 - G_2)\\ (G_1 - G_2) & -(G_1 + G_2)\end{pmatrix}.$$

Most probably, $\partial_{\boldsymbol{\theta}\boldsymbol{\theta}}\mathcal{F}$ is positive definite but $\partial_{\boldsymbol{\nu}\boldsymbol{\nu}}\mathcal{F}$ is always negative, and $\mathcal{F}$ is generally saddle at equilibrium.

### 3.4 Perturbation Analysis

For the following discussion, we define a distribution $p(\boldsymbol{x}; \boldsymbol{\theta}, \boldsymbol{v})$ as

$$p(\boldsymbol{x}; \boldsymbol{\theta}, \boldsymbol{v}) = \exp\left(c_0(\boldsymbol{x}) + \boldsymbol{\theta}\cdot\boldsymbol{x} + \boldsymbol{v}\cdot\boldsymbol{c}(\boldsymbol{x}) - \varphi(\boldsymbol{\theta}, \boldsymbol{v})\right), \quad \boldsymbol{v} = (v^1, v^2)^T,$$

$$\varphi(\boldsymbol{\theta}, \boldsymbol{v}) = \ln\sum_{\boldsymbol{x}}\exp\left(c_0(\boldsymbol{x}) + \boldsymbol{\theta}\cdot\boldsymbol{x} + \boldsymbol{v}\cdot\boldsymbol{c}(\boldsymbol{x})\right), \quad \boldsymbol{c}(\boldsymbol{x}) \stackrel{\text{def}}{=} (c_1(\boldsymbol{x}), c_2(\boldsymbol{x}))^T.$$

This distribution includes $p_0(\boldsymbol{x}; \boldsymbol{\theta})$ $(\boldsymbol{v} = \mathbf{0})$, $p(\boldsymbol{x}|\tilde{\boldsymbol{x}}, \tilde{\boldsymbol{y}}_1, \tilde{\boldsymbol{y}}_2)$ $(\boldsymbol{\theta} = \mathbf{0}, \boldsymbol{v} = \mathbf{1})$, and $p_r(\boldsymbol{x}; \boldsymbol{\xi})$ $(\boldsymbol{\theta} = \boldsymbol{\xi}, \boldsymbol{v} = \boldsymbol{e}_r)$, where $\mathbf{1} = (1, 1)^T$, $\boldsymbol{e}_1 = (1, 0)^T$, and $\boldsymbol{e}_2 = (0, 1)^T$. The expectation parameter $\boldsymbol{\eta}(\boldsymbol{\theta}, \boldsymbol{v})$ is defined as,

$$\boldsymbol{\eta}(\boldsymbol{\theta}, \boldsymbol{v}) = \partial_{\boldsymbol{\theta}}\varphi(\boldsymbol{\theta}, \boldsymbol{v}) = \sum_{\boldsymbol{x}}\boldsymbol{x}p(\boldsymbol{x}; \boldsymbol{\theta}, \boldsymbol{v}).$$

Let us consider $M(\boldsymbol{\theta}^*)$, where every distribution $p(\boldsymbol{x}; \boldsymbol{\theta}, \boldsymbol{v})\in M(\boldsymbol{\theta}^*)$ has the same expectation parameter, that is, $\boldsymbol{\eta}(\boldsymbol{\theta}, \boldsymbol{v}) = \boldsymbol{\eta}(\boldsymbol{\theta}^*)$ holds. Here, we define, $\boldsymbol{\eta}(\boldsymbol{\theta}^*) = \boldsymbol{\eta}(\boldsymbol{\theta}^*, \mathbf{0})$. From the Taylor expansion, we have,

$$\eta_i(\boldsymbol{\theta}, \boldsymbol{v}) = \eta_i(\boldsymbol{\theta}^*) + \sum_j \partial_j\eta_i(\boldsymbol{\theta}^*)\Delta\theta^j + \sum_r \partial_r\eta_i(\boldsymbol{\theta}^*)v^r + \frac{1}{2}\sum_{r,s}\partial_r\partial_s\eta_i(\boldsymbol{\theta}^*)v^r v^s$$

$$+ \sum_{j,r}\partial_r\partial_j\eta_i(\boldsymbol{\theta}^*)v^r\Delta\theta^j + \frac{1}{2}\sum_{k,l}\partial_k\partial_l\eta_i(\boldsymbol{\theta}^*)\Delta\theta^k\Delta\theta^l + O(\|\boldsymbol{v}\|^3) + O(\|\Delta\boldsymbol{\theta}\|^3). \tag{8}$$

The indexes $\{i, j, k, l\}$ are for $\boldsymbol{\theta}$, $\{r, s\}$ are for $\boldsymbol{v}$, and $\Delta\boldsymbol{\theta} \stackrel{\text{def}}{=} \boldsymbol{\theta} - \boldsymbol{\theta}^*$. After adding some definitions, that is, $\eta_i(\boldsymbol{\theta}, \boldsymbol{v}) = \eta_i(\boldsymbol{\theta}^*)$, and $\partial_j \eta_i(\boldsymbol{\theta}^*) = g_{ij}(\boldsymbol{\theta}^*)$, where $\{g_{ij}\}$ is the Fisher information matrix of $p(\boldsymbol{x}; \boldsymbol{\theta}^*, \mathbf{o})$ which is a diagonal matrix, we substitute $\Delta\theta^i$ with function of $v^r$ up to its 2nd order, and neglect the higher orders of $v^r$. And we have,

$$\Delta\theta^i \simeq -g^{ii}\sum_r A_{ir}v^r - \frac{g^{ii}}{2}\sum_{r,s}\Big(\partial_r - \sum_k g^{kk}A_{kr}\partial_k\Big)\Big(\partial_s - \sum_j g^{jj}A_{js}\partial_j\Big)\eta_i(\boldsymbol{\theta}^*)v^r v^s,$$

(9)

where, $g^{ii} = 1/g_{ii}$, and $A_{ir} = \partial_{v^r}\eta_i(\boldsymbol{\theta}^*)$.

Let $\boldsymbol{v} = \boldsymbol{e}_1$, and since $p(\boldsymbol{x}; \boldsymbol{\theta}, \boldsymbol{e}_1) = p_1(\boldsymbol{x}; \boldsymbol{\theta}) \in M(\boldsymbol{\theta}^*)$ holds, $\boldsymbol{\theta} = \boldsymbol{\xi}_2^*$ and $\Delta\boldsymbol{\theta} = \boldsymbol{\xi}_2^* - \boldsymbol{\theta}^* = -\boldsymbol{\xi}_1^*$. Also when we put $\boldsymbol{v} = \boldsymbol{e}_2$, $\Delta\boldsymbol{\theta} = -\boldsymbol{\xi}_2^*$ holds. From eq.(9), we have the following result,

$$-\xi_r^{i,*} \simeq -g^{ii}A_{ir} - \frac{g^{ii}}{2}\Big(\partial_r - \sum_k g^{kk}A_{kr}\partial_k\Big)\Big(\partial_r - \sum_j g^{jj}A_{jr}\partial_j\Big)\eta_i(\boldsymbol{\theta}^*). \qquad (10)$$

Next, let $\boldsymbol{v} = \mathbf{1}$, and we consider $p(\boldsymbol{x}; \bar{\boldsymbol{\theta}}, \mathbf{1}) \in M(\boldsymbol{\theta}^*)$, where $\bar{\boldsymbol{\theta}}$ is the parameter which satisfies this equation. Since $p(\boldsymbol{x}; \mathbf{o}, \mathbf{1}) = p(\boldsymbol{x}|\tilde{\boldsymbol{x}}, \tilde{\boldsymbol{y}}_1, \tilde{\boldsymbol{y}}_2)$ is not necessarily included in $M(\boldsymbol{\theta}^*)$, $\bar{\boldsymbol{\theta}}$ is generally not equal to $\mathbf{o}$. From eq.(9),

$$\bar{\theta}^i - \theta^{i,*} \simeq -g^{ii}\sum_r A_{ir} - \frac{g^{ii}}{2}\sum_r\Big(\partial_r - \sum_k g^{kk}A_{kr}\partial_k\Big)\Big(\partial_r - \sum_j g^{jj}A_{jr}\partial_j\Big)\eta_i(\boldsymbol{\theta}^*).$$

From the condition $\boldsymbol{\theta}^* = \boldsymbol{\xi}_1^i + \boldsymbol{\xi}_2^i$ and eq.(10), we have the following approximation,

$$\bar{\theta}^i \simeq -\frac{g^{ii}}{2}\sum_{r\neq s}\Big(\partial_r - \sum_k g^{kk}A_{kr}\partial_k\Big)\Big(\partial_s - \sum_j g^{jj}A_{js}\partial_j\Big)\eta_i(\boldsymbol{\theta}^*).$$

This result gives the approximation accuracy of the BP decoding. Let the true belief be $\boldsymbol{\eta}_{MPM}$, and we evaluate the difference between $\boldsymbol{\eta}_{MPM}$ and $\boldsymbol{\eta}(\boldsymbol{\theta}^*)$ on $M_0$. The result is summarized in the following theorem.

**Theorem 5.** *The true expectation of $\boldsymbol{x}$, which is $\boldsymbol{\eta}_{MPM} = \boldsymbol{\eta}(\mathbf{o}, \mathbf{1})$, is approximated as,*

$$\boldsymbol{\eta}_{MPM} \simeq \boldsymbol{\eta}(\boldsymbol{\theta}^*) + \frac{1}{2}\sum_{r\neq s}\Big(\partial_r - \sum_k g^{kk}A_{kr}\partial_k\Big)\Big(\partial_s - \sum_j g^{jj}A_{js}\partial_j\Big)\boldsymbol{\eta}(\boldsymbol{\theta}^*). \qquad (11)$$

*Where $\boldsymbol{\eta}(\boldsymbol{\theta}^*)$ is the solution of the turbo decoding.* $\qquad\qquad\square$

Equation (11) is related to the $m$–embedded–curvature of $E(\boldsymbol{\theta}^*)$ (Fig.3). The result can be extended to general case where $K \geq 3$ [3, 8].

## 4 Discussion

We have shown a new framework for understanding and analyzing the belief propagation decoder.

Since the BN of turbo codes is loopy, we don't have enough theoretical results for BP algorithm, while a lot of experiments show that it works surprisingly well in such cases. The mystery of the BP decoders is summarized in 2 points, the approximation accuracy and the convergence property.

Our results elucidate the mathematical background of the BP decoding algorithm. The information geometrical structure of the equilibrium is summarized in Theorem 2. It shows

the $e$–flat submanifold $E(\boldsymbol{\theta}^*)$ plays an important role. Furthermore, Theorem 5 shows that the relation between $E(\boldsymbol{\theta}^*)$ and the $m$–flat submanifold $M(\boldsymbol{\theta}^*)$ causes the decoding error, and the principal component of the error is the curvature of $E(\boldsymbol{\theta}^*)$. Since the curvature strongly depends on the codeword, we can control it by the encoder design. This shows a room for improvement of the "near optimum error correcting code"[2].

For the convergent property, we have shown the energy function, which is known as Bethé free energy[4, 9]. Unfortunately, the fixed point of the turbo decoding algorithm is generally a saddle of the function, which makes further analysis difficult. We have only shown a local stability condition, and the global property is one of our future works.

This paper gives a first step to the information geometrical understanding of the belief propagation decoder. The main results are for the turbo decoding, but the mechanism is common with wider class, and the framework is valid for them. We believe further study in this direction will lead us to better understanding and improvements of these methods.

## Acknowledgments

We thank Chiranjib Bhattacharyya who gave us the opportunity to face this problem. We are also grateful to Yoshiyuki Kabashima and Motohiko Isaka for useful discussions.

## References

[1] S. Amari and H. Nagaoka. (2000) *Methods of Information Geometry*, volume 191 of *Translations of Mathematical Monographs*. American Mathematical Society.

[2] C. Berrou and A. Glavieux. (1996) Near optimum error correcting coding and decoding: Turbo-codes. *IEEE Transactions on Communications*, 44(10):1261–1271.

[3] S. Ikeda, T. Tanaka, and S. Amari. (2001) Information geometry of turbo codes and low-density parity-check codes. submitted to IEEE transaction on Information Theory.

[4] Y. Kabashima and D. Saad. (2001) The TAP approach to intensive and extensive connectivity systems. In M. Opper and D. Saad, editors, *Advanced Mean Field Methods – Theory and Practice*, chapter 6, pages 65–84. The MIT Press.

[5] R. J. McEliece, D. J. C. MacKay, and J.-F. Cheng. (1998) Turbo decoding as an instance of Pearl's "belief propagation" algorithm. *IEEE Journal on Selected Areas in Communications*, 16(2):140–152.

[6] T. J. Richardson. (2000) The geometry of turbo-decoding dynamics. *IEEE Transactions on Information Theory*, 46(1):9–23.

[7] T. Tanaka. (2001) Information geometry of mean-field approximation. In M. Opper and D. Saad, editors, *Advanced Mean Field Methods – Theory and Practice*, chapter 17, pages 259–273. The MIT Press.

[8] T. Tanaka, S. Ikeda, and S. Amari. (2002) Information-geometrical significance of sparsity in Gallager codes. in T. G. Dietterich *et al.* (eds.), *Advances in Neural Information Processing Systems*, vol. 14 (this volumn), The MIT Press.

[9] J. S. Yedidia, W. T. Freeman, and Y. Weiss. (2001) Bethe free energy, Kikuchi approximations, and belief propagation algorithms. Technical Report TR2001–16, Mitsubishi Electric Research Laboratories.
